# Sparse Metric Learning via Smooth Optimization

**Yiming Ying†, Kaizhu Huang‡, and Colin Campbell†**
†Department of Engineering Mathematics, University of Bristol,
Bristol BS8 1TR, United Kingdom
‡National Laboratory of Pattern Recognition, Institute of Automation,
The Chinese Academy of Sciences, 100190 Beijing, China

## Abstract

In this paper we study the problem of learning a low-rank (sparse) distance matrix. We propose a novel metric learning model which can simultaneously conduct dimension reduction and learn a distance matrix. The sparse representation involves a mixed-norm regularization which is non-convex. We then show that it can be equivalently formulated as a convex saddle (min-max) problem. From this saddle representation, we develop an efficient smooth optimization approach [17] for sparse metric learning, although the learning model is based on a non-differentiable loss function. Finally, we run experiments to validate the effectiveness and efficiency of our sparse metric learning model on various datasets.

## 1 Introduction

For many machine learning algorithms, the choice of a distance metric has a direct impact on their success. Hence, choosing a good distance metric remains a challenging problem. There has been much work attempting to exploit a distance metric in many learning settings, e.g. [8, 9, 10, 12, 20, 22, 23, 25]. These methods have successfully indicated that a good distance metric can significantly improve the performance of $k$-nearest neighbor classification and $k$-means clustering, for example.

A good choice of a distance metric generally preserves the *distance structure* of the data: the distance between examples exhibiting *similarity* should be relatively smaller, in the transformed space, than between examples exhibiting *dissimilarity*. For supervised classification, the label information indicates whether the pair set is in the same class (similar) or in the different classes (dissimilar). In semi-supervised clustering, the side information conveys the information that a pair of samples are similar or dissimilar to each other. Since it is very common that the presented data is contaminated by noise, especially for high-dimensional datasets, a good distance metric should also be minimally influenced by noise. In this case, a low-rank distance matrix would produce a better generalization performance than non-sparse counterparts and provide a much faster and efficient distance calculation for test samples. Hence, a good distance metric should also pursue dimension reduction during the learning process.

In this paper we present a novel approach to learn a low-rank (sparse) distance matrix. We first propose in Section 2 a novel metric learning model for estimating the linear transformation (equivalently distance matrix) that combines and retains the advantages of existing methods [8, 9, 12, 20, 22, 23, 25]. Our method can simultaneously conduct dimension reduction and learn a low-rank distance matrix. The sparse representation is realized by a mixed-norm regularization used in various learning settings [1, 18, 21]. We then show that this non-convex mixed-norm regularization framework is equivalent to a convex saddle (min-max) problem. Based on this equivalent representation, we develop, in Section 3, Nesterov's smooth optimization approach [16, 17] for sparse metric learning using smoothing approximation techniques, although the learning model is based on a non-differentiable loss function. In Section 4, we demonstrate the effectiveness and efficiency of our sparse metric learning model with experiments on various datasets.

## 2 Sparse Distance Matrix Learning Model

We begin by introducing necessary notation. Let $\mathbb{N}_n = \{1, 2, \ldots, n\}$ for any $n \in \mathbb{N}$. The space of symmetric $d$ times $d$ matrices will be denoted by $\mathcal{S}^d$. If $S \in \mathcal{S}^d$ is positive definite, we write it as $S \succeq 0$. The cone of positive semi-definite matrices is denoted by $\mathcal{S}^d_+$ and denote by $\mathcal{O}^d$ the set of $d$ times $d$ orthonormal matrices. For any $X, Y \in \mathbb{R}^{d \times q}$, $\langle X, Y \rangle := \mathbf{Tr}(X^\top Y)$ where $\mathbf{Tr}(\cdot)$ denotes the trace of a matrix. The standard Euclidean norm is denoted by $\| \cdot \|$. Denote by $\mathbf{z} := \{(x_i, y_i) : i \in \mathbb{N}_n\}$ a training set of $n$ labeled examples with input $x_i = (x_i^1, \ldots, x_i^d) \in \mathbb{R}^d$, class label $y_i$ (not necessary binary) and let $x_{ij} = x_i - x_j$.

Let $P = (P_{\ell k})_{\ell, k \in \mathbb{N}_d} \in R^{d \times d}$ be a *transformation matrix*. Denote by $\hat{x}_i = P x_i$ for any $i \in \mathbb{N}_n$ and by $\hat{\mathbf{x}} = \{\hat{x}_i : i \in \mathbb{N}_n\}$ the *transformed data matrix*. The linear transformation matrix $P$ induces a distance matrix $M = P^\top P$ which defines a distance between $x_i$ and $x_j$ given by

$$d_M(x_i, x_j) = (x_i - x_j)^\top M (x_i - x_j).$$

Our sparse metric learning model is based on two principal hypotheses: 1) a good choice of distance matrix $M$ should preserve the distance structure, i.e. the distance between similar examples should be relatively smaller than between dissimilar examples; 2) a good distance matrix should also be able to effectively remove noise leading to dimension reduction.

For the first hypothesis, the distance structure in the transformed space can be specified, for example, by the following constraints: $\|P(x_j - x_k)\|^2 \geq \|P(x_i - x_j)\|^2 + 1, \forall (x_i, x_j) \in \mathcal{S}$ and $(x_j, x_k) \in \mathcal{D}$, where $\mathcal{S}$ denotes the similarity pairs and $\mathcal{D}$ denotes the dissimilarity pairs based on the label information. Equivalently,

$$\|\hat{x}_j - \hat{x}_k\|^2 \geq \|\hat{x}_i - \hat{x}_j\|^2 + 1, \forall (x_i, x_j) \in \mathcal{S} \text{ and } (x_j, x_k) \in \mathcal{D}. \tag{1}$$

For the second hypothesis, we use a sparse regularization to give a sparse solution. This regularization ranges from element-sparsity for variable selection to a low-rank matrix for dimension reduction [1, 2, 3, 13, 21]. In particular, for any $\ell \in \mathbb{N}_d$, denote the $\ell$-th row vector of $P$ by $P_\ell$ and $\|P_\ell\| = (\sum_{k \in \mathbb{N}_d} P_{\ell k}^2)^{\frac{1}{2}}$. If $\|P_\ell\| = 0$ then the $\ell$-th variable in the transformed space becomes zero, i.e. $x_i^\ell = P_\ell x_i = 0$ which means that $\|P_\ell\| = 0$ has the effect of eleminating $\ell$-th variable. Motivated by the above observation, a direct way would be to enforce a $L^1$-norm across the vector $(\|P_1\|, \ldots, \|P_d\|)$, i.e. $\sum_{\ell \in \mathbb{N}_d} \|P_\ell\|$. This $L^1$-regularization yields row-vector (feature) sparsity of $\hat{\mathbf{x}}$ which plays the role of feature selection. Let $W = P^\top P = (W_1, \ldots, W_d)$ and we can easily show that

$$W_\ell \equiv 0 \iff P_\ell \equiv 0.$$

Motivated by this observation, instead of $L^1$-regularization over vector $(\|P_1\|, \ldots, \|P_d\|)$ we can enforce $L^1$-norm regularization across the vector $(\|W_1\|, \ldots, \|W_d\|)$. However, a low-dimensional projected space $\hat{\mathbf{x}}$ does not mean that its row-vector (feature) should be sparse. Ideally, we expect that the principal component of $\hat{\mathbf{x}}$ can be sparse. Hence, we introduce an extra orthonormal transformation $U \in \mathcal{O}^d$ and let $\hat{x}_i = PU x_i$. Denote a set of triplets $\mathcal{T}$ by

$$\mathcal{T} = \{\tau = (i, j, k) : i, j, k \in \mathbb{N}_n , (x_i, x_j) \in \mathcal{S} \text{ and } (x_j, x_k) \in \mathcal{D}\}. \tag{2}$$

By introducing slack variables $\xi$ in constraints (1), we propose the following sparse (low-rank) distance matrix learning formulation:

$$\begin{aligned} \min_{U \in \mathcal{O}^d} \min_{W \in \mathcal{S}^d_+} \quad & \sum_\tau \xi_\tau + \gamma \|W\|_{(2,1)}^2 \\ \text{s.t.} \quad & 1 + x_{ij}^\top U^\top W U x_{ij} \leq x_{kj}^\top U^\top W U x_{kj} + \xi_\tau, \\ & \xi_\tau \geq 0, \ \forall \tau = (i, j, k) \in \mathcal{T}, \text{ and } W \in \mathcal{S}^d_+. \end{aligned} \tag{3}$$

where $\|W\|_{(2,1)} = \sum_\ell (\sum_k w_{k\ell}^2)^{\frac{1}{2}}$ denotes the $(2, 1)$-norm of $W$. A similar mixed $(2, 1)$-norm regularization was used in [1, 18] for multi-task learning and multi-class classification to learn the sparse representation shared across different tasks or classes.

### 2.1 Equivalent Saddle Representation

We now turn our attention to an equivalent saddle (min-max) representation for sparse metric learning (3) which is essential for developing optimization algorithms in the next section. To this end, we need the following lemma which develops and extends a similar version in multi-task learning [1, 2] to the case of learning a positive semi-definite distance matrix.

**Lemma 1.** *Problem (3) is equivalent to the following convex optimization problem*

$$\min_{M \succeq 0} \sum_{\tau=(i,j,k)\in\mathcal{T}} (1 + x_{ij}^\top M x_{ij} - x_{kj}^\top M x_{kj})_+ + \gamma(\mathbf{Tr}(M))^2 \tag{4}$$

*Proof.* Let $M = UWU^\top$ in equation (3) and then $W = U^\top M U$. Hence, (3) is reduced to the following

$$\min_{M \in \mathcal{S}_+^d} \min_{U \in \mathscr{O}^d} \sum_\tau \xi_\tau + \gamma \|U^\top M U\|_{(2,1)}^2 \tag{5}$$

$$\text{s.t. } x_{ij}^\top M x_{ij} \leq x_{kj}^\top M x_{kj} + \xi_\tau,$$

$$\xi_\tau \geq 0 \ \forall \tau = (i,j,k) \in \mathcal{T}, \text{ and } M \in \mathcal{S}_+^d.$$

Now, for any fixed $M$ in equation (5), by the eigen-decomposition of $M$ there exists $\widetilde{U} \in \mathscr{O}^d$ such that $M = \widetilde{U}^\top \lambda(M) \widetilde{U}$. Here, the diagonal matrix $\lambda(M) = \text{diag}(\lambda_1, \lambda_2, \ldots, \lambda_d)$ where $\lambda_i$ is the $i$-th eigenvalue of $M$. Let $V = \widetilde{U}U \in \mathscr{O}^d$, and then we have $\min_{U\in\mathscr{O}^d} \|U^\top M U\|_{(2,1)} = \min_{U\in\mathscr{O}^d} \|(\widetilde{U}U)^\top \lambda(M)\widetilde{U}U\|_{(2,1)} = \min_{V\in\mathscr{O}^d} \|V^\top \lambda(M)V\|_{(2,1)}$. Observe that

$$\|V^\top \lambda(M)V\|_{(2,1)} = \sum_i (\sum_j (\sum_k V_{ki}\lambda_k V_{kj})^2)^{\frac{1}{2}}$$
$$= \sum_i (\sum_{k,k'} (\sum_j V_{ki} V_{k'i})\lambda_k V_{kj}\lambda_{k'} V_{k'j})^{\frac{1}{2}} = \sum_i (\sum_k \lambda_k^2 V_{ki}^2)^{\frac{1}{2}} \tag{6}$$

where, in the last equality, we use the fact that $V \in \mathscr{O}^d$, i.e. $\sum_j V_{kj}V_{k'j} = \delta_{kk'}$. Applying Cauchy-Schwartz's inequality implies that $\sum_k \lambda_k V_{ki}^2 \leq (\sum_k \lambda_k^2 V_{ki}^2)^{\frac{1}{2}} (\sum_k V_{ki}^2)^{\frac{1}{2}} = (\sum_k \lambda_k^2 V_{ki}^2)^{\frac{1}{2}}$. Putting this back into (6) yields $\|V^\top \lambda(M)V\|_{(2,1)} \geq \sum_i \sum_k \lambda_k V_{ki}^2 = \sum_k \lambda_k = \mathbf{Tr}(M)$, where we use the fact $V \in \mathscr{O}^d$ again. However, if we select $V$ to be identity matrix $\mathbf{I}_d$, $\|V^\top \lambda(M)V\|_{(2,1)} = \mathbf{Tr}(M)$. Hence, $\min_{U\in\mathscr{O}^d} \|U^\top M U\|_{(2,1)} = \min_{V\in\mathscr{O}^d} \|V^\top \lambda(M)V\|_{(2,1)} = \mathbf{Tr}(M)$. Putting this back into equation (5) the result follows. $\square$

From the above lemma, we are ready to present an equivalent saddle (min-max) representation of problem (3). First, let $\mathcal{Q}_1 = \{u_\tau : \tau \in \mathcal{T}, 0 \leq u_\tau \leq 1\}$ and $\mathcal{Q}_2 = \{M \in \mathcal{S}_+^d : \mathbf{Tr}(M) \leq \sqrt{T/\gamma}\}$ where $T$ is the cardinality of triplet set $\mathcal{T}$ i.e. $T = \#\{\tau \in \mathcal{T}\}$.

**Theorem 1.** *Problem (4) is equivalent to the following saddle representation*

$$\min_{u\in\mathcal{Q}_1} \max_{M\in\mathcal{Q}_2} \left\{ \langle \sum_{\tau=(i,j,k)\in\mathcal{T}} u_\tau (x_{jk}x_{jk}^\top - x_{ij}x_{ij}^\top), M \rangle - \gamma(\mathbf{Tr}(M))^2 \right\} - \sum_{t\in\mathcal{T}} u_\tau \tag{7}$$

*Proof.* Suppose that $M^*$ is an optimal solution of problem (4). By its definition, there holds $\gamma(\mathbf{Tr}(M^*))^2 \leq \sum_{\tau\in\mathcal{T}} (1 + x_{kj}^\top M x_{ik} - x_{kj}^\top M x_{kj})_+ + \gamma(\mathbf{Tr}(M))^2$ for any $M \succeq 0$. Letting $M = 0$ yields that $\mathbf{Tr}(M^*) \leq \sqrt{T/\gamma}$. Hence, problem (4) is identical to

$$\min_{M\in\mathcal{Q}_2} \sum_{\tau=(i,j,k)\in\mathcal{T}} (1 + x_{ij}^\top M x_{ij} - x_{kj}^\top M x_{kj})_+ + \gamma(\mathbf{Tr}(M))^2. \tag{8}$$

Observe that $s_+ = \max\{0, s\} = \max_\alpha\{s\alpha : 0 \leq \alpha \leq 1\}$. Consequently, the above equation can be written as $\min_{M\in\mathcal{Q}_2} \max_{0\leq u \leq 1} \sum_{\tau\in\mathcal{T}} u_\tau(1 + x_{kj}^\top M x_{ik} - x_{ij}^\top M x_{ij}) + \gamma(\mathbf{Tr}(M))^2$. By the min-max theorem (e.g. [5]), the above problem is equivalent to $\min_{u\in\mathcal{Q}_1} \max_{M\in\mathcal{Q}_2} \left\{ \sum_{\tau\in\mathcal{T}} u_\tau(-x_{ij}^\top M x_{ij} + x_{jk}^\top M x_{jk}) - \gamma(\mathbf{Tr}(M))^2 \right\} - \sum_{\tau\in\mathcal{T}} u_t$. Combining this with the fact that $x_{jk}^\top M x_{jk} - x_{ij}^\top M x_{ij} = \langle x_{jk}x_{jk}^\top - x_{ij}x_{ij}^\top, M \rangle$ completes the proof of the theorem. $\square$

## 2.2 Related Work

There is a considerable amount of work on metric learning. In [9], an information-theoretic approach to metric learning (ITML) is developed which equivalently transforms the metric learning problem

to that of learning an optimal Gaussian distribution with respect to an relative entropy. The method of Relevant Component analysis (RCA)[7] attempts to find a distance metric which can minimize the covariance matrix imposed by the equivalence constraints. In [25], a distance metric for $k$-means clustering is then learned to shrink the averaged distance within the similar set while enlarging the average distance within the dissimilar set simultaneously. All the above methods generally do not yield sparse solutions and only work within their special settings. Maximally Collapsing Metric Learning (MCML) tries to map all points in a same class to a single location in the feature space via a stochastic selection rule. There are many other metric learning approaches in either unsupervised or supervised learning setting, see [26] for a detailed review. We particularly mention the following work which is more related to our sparse metric learning model (3).

• Large Margin Nearest Neighbor (LMNN) [23, 24]: LMNN aims to explore a large margin nearest neighbor classifier by exploiting nearest neighbor samples as side information in the training set. Specifically, let $\mathcal{N}_k(x)$ denotes the $k$-nearest neighbor of sample $x$ and define the similar set $\mathcal{S} = \{(x_i, x_j) : x_i \in \mathcal{N}(x_j), y_i = y_j\}$ and $\mathcal{D} = \{(x_j, x_k) : x_k \in \mathcal{N}(x_j), y_k \neq y_j\}$. Then, recall that the triplet set $\mathcal{T}$ is given by equation (2), the framework LMNN can be rewritten as the following:

$$\min_{M \succeq 0} \sum_{\tau=(i,j,k) \in \mathcal{T}} (1 + x_{ij}^\top M x_{ij} - x_{kj}^\top M x_{kj})_+ + \gamma \mathbf{Tr}(\mathcal{C}M) \tag{9}$$

where the covariance matrix $\mathcal{C}$ over the similar set $\mathcal{S}$ is defined by $\mathcal{C} = \sum_{(x_i,x_j) \in \mathcal{S}} (x_i - x_j)(x_i - x_j)^\top$. From the above reformulation, we see that LMNN also involves a sparse regularization term $\mathbf{Tr}(\mathcal{C}M)$. However, the sparsity of $\mathcal{C}M$ does not imply the sparsity of $M$, see the discussion in the experimental section. Large Margin Component Analysis (LMCA) [22] is designed for conducting classification and dimensionality reduction simultaneously. However, LMCA controls the sparsity by directly specifying the dimensionality of the transformation matrix and it is an extended version of LMNN. In practice, this low dimensionality is tuned by *ad hoc* methods such as cross-validation.

• Sparse Metric Learning via Linear Programming (SMLlp) [20]: the spirit of this approach is closer to our method where the following sparse framework was proposed:

$$\min_{M \succeq 0} \sum_{t=(i,j,k) \in \mathcal{T}} (1 + x_{ij}^\top M x_{ij} - x_{kj}^\top M x_{kj})_+ + \gamma \sum_{\ell, k \in \mathbb{N}_d} |M_{\ell k}| \tag{10}$$

However, the above 1-norm term $\sum_{\ell, k \in \mathbb{N}_d} |M_{\ell k}|$ can only enforce the element sparsity of $M$. The learned sparse model would not generate an appropriate low-ranked principal matrix $M$ for metric learning. In order to solve the above optimization problem, [10] further proposed to restrict $M$ to the space of diagonal dominance matrices: a small subspace of the positive semi-definite cone. Such a restriction would only result in a sub-optimal solution, although the final optimization is an efficient linear programming problem.

## 3 Smooth Optimization Algorithms

Nesterov [17, 16] developed an efficient smooth optimization method for solving convex programming problems of the form $\min_{x \in Q} f(x)$ where $Q$ is a bounded closed convex set in a finite-dimensional real vector space $E$. This smooth optimization usually requires $f$ to be differentiable with Lipschitz continuous gradient and it has an optimal convergence rate of $\mathcal{O}(1/t^2)$ for smooth problems where $t$ is the iteration number. Unfortunately, we can not directly apply the smooth optimization method to problem (4) since the hinge loss there is not continuously differentiable. Below we show the smooth approximation method [17] can be approached through the saddle representation (7).

### 3.1 Nesterov's Smooth Approximation Approach

We briefly review Nesterov's approach [17] in the setting of a general min-max problem using smoothing techiniques. To this end, we introduce some useful notation. Let $\mathcal{Q}_1$ (resp. $\mathcal{Q}_2$) be nonempty convex compact sets in finite-dimensional real vector spaces $E_1$ (resp. $E_2$) endowed with norm $\| \cdot \|_1$ (resp. $\| \cdot \|_2$). Let $E_2^*$ be the dual space of $E_2$ with standard norm defined, for any $s \in E_2^*$, by $\|s\|_2^* = \max\{\langle s, x \rangle_2 : \|x\|_2 = 1\}$, where the scalar product $\langle \cdot, \cdot \rangle_2$ denotes the value of $s$ at $x$. Let $A : E_1 \to E_2^*$ be a linear operator. Its adjoint operator $A^* : E_2 \to E_1^*$ is defined,

| **Smooth Optimization Algorithm for Sparse Metric Learning (SMLsm)** |
|---|
| 1. Let $\varepsilon > 0$, $t = 0$ and initialize $u^{(0)} \in \mathcal{Q}_1$, $M^{(-1)} = 0$ and let $L = \frac{1}{2\mu} \sum_{\tau \in \mathcal{T}} \|X_\tau\|_2^2$ |
| 2. Compute $M_\mu(u^{(t)})$ and $\nabla\phi_\mu(u^{(t)}) = (-1 + \langle X_\tau, M_\mu(u^{(t)}) \rangle : \tau \in \mathcal{T})$ and let $M^{(t)} = \frac{t}{t+2} M^{(t-1)} + \frac{2}{t+2} M_\mu(u^t)$ |
| 3. Compute $z^{(t)} = \arg\min_{z \in \mathcal{Q}_1} \left\{ \frac{L}{2}\|u^{(t)} - z\|^2 + \nabla\phi_\mu(u^{(t)})^\top (z - u^{(t)}) \right\}$ |
| 4. Compute $v^{(t)} = \arg\min_{v \in \mathcal{Q}_1} \left\{ \frac{L}{2}\|u^{(0)} - v\|^2 + \sum_{i=0}^{t}(\frac{i+1}{2})(\phi_\mu(u^{(i)}) + \nabla\phi_\mu(u^{(i)})^\top (v - u^{(i)})) \right\}$ |
| 5. Set $u^{(t+1)} = \frac{2}{t+3} v^{(t)} + \frac{t+1}{t+3} z^{(t)}$ |
| 6. Set $t \leftarrow t + 1$. Go to step 2 until the stopping criterion less than $\varepsilon$ |

Table 1: Pseudo-code of first order Nesterov's method

for any $x \in E_2$ and $u \in E_1$, by $\langle Au, x \rangle_2 = \langle A^* x, u \rangle_1$. The norm of such a operator is defined by $\|A\|_{1,2} = \max_{x,u} \{ \langle Au, x \rangle_2 : \|x\|_2 = 1, \|u\|_1 = 1 \}$.

Now, the min-max problem considered in [17, Section 2] has the following special structure:

$$\min_{u \in \mathcal{Q}_1} \left\{ \phi(u) = \widehat{\phi}(u) + \max\{ \langle Au, x \rangle_2 - \hat{f}(x) : x \in \mathcal{Q}_2 \} \right\}. \tag{11}$$

Here, $\widehat{\phi}(u)$ is assumed to be continuously differentiable and convex with Lipschitz continuous gradient and $\hat{f}(x)$ is convex and differentiable. The above min-max problem is usually not smooth and Nesterov [17] proposed a smoothing approximation approach to solve the above problem:

$$\min_{u \in \mathcal{Q}_1} \left\{ \phi_\mu(u) = \widehat{\phi}(u) + \max\{ \langle Au, x \rangle_2 - \hat{f}(x) - \mu d_2(x) : x \in \mathcal{Q}_2 \} \right\}. \tag{12}$$

Here, $d_2(\cdot)$ is a continuous *proxy-function*, strongly convex on $\mathcal{Q}_2$ with some convexity parameter $\sigma_2 > 0$ and $\mu > 0$ is a small smoohting parameter. Let $x_0 = \arg\min_{x \in \mathcal{Q}_2} d_2(x)$. Without loss of generality, assume $d_2(x_0) = 0$. The strong convexity of $d_2(\cdot)$ with parameter $\sigma_2$ means that $d_2(x) \geq \frac{1}{2}\sigma_2\|x - x_0\|_2^2$. Since $d_2(\cdot)$ is strongly convex, the solution of the maximization problem $\hat{\phi}_\mu(u) := \max\{ \langle Au, x \rangle_2 - \hat{f}(x) - \mu d_2(x) : x \in \mathcal{Q}_2 \}$ is unique and differentiable, see [6, Theorem 4.1]. Indeed, it was established in [17, Theorem 1] that the gradient of $\phi_\mu$ is given by

$$\nabla\hat{\phi}_\mu(u) = A^* x_\mu(u) \tag{13}$$

and it has a Lipschitz constant $L = \frac{\|A\|_{1,2}^2}{\mu\sigma_2}$, i.e. $\|A^* x_\mu(u_1) - A^* x_\mu(u_2)\|_1^* \leq \frac{\|A\|_{1,2}^2}{\mu\sigma_2}\|u_1 - u_2\|_1$. Hence, the proxy-function $d_2$ can be regarded as a generalized Moreau-Yosida regularization term to smooth out the objective function.

As mentioned above, function $\phi_\mu$ in problem (12) is differentiable with Lipschitz continuous gradients. Hence, we can apply the optimal smooth optimization scheme [17, Section 3] to the smooth approximate problem (12). The optimal scheme needs another proxy-function $d(u)$ associated with $\mathcal{Q}_1$. Assume that $d(u_0) = \min_{u \in \mathcal{Q}_1} d(u) = 0$ and it has convexity parameter $\sigma$ i.e. $d(u) \geq \frac{1}{2}\sigma\|u - u_0\|_1$. For this special problem (12), the primal solution $u^* \in \mathcal{Q}_1$ and dual solution $x^* \in \mathcal{Q}_2$ can be simultaneously obtained, see [17, Theorem 3]. Below, we will apply this general scheme to solve the min-max representation (7) of the sparse metric learning problem (3), and hence solves the original problem (4).

### 3.2 Smooth Optimization Approach for Sparse Metric Learning

We now turn our attention to developing a smooth optimization approach for problem (4). Our main idea is to connect the saddle representation (7) in Theorem 1 with the special formulation (11).

To this end, firstly let $E_1 = \mathbb{R}^T$ with standard Euclidean norm $\|\cdot\|_1 = \|\cdot\|$ and $E_2 = \mathcal{S}^d$ with Frobenius norm defined, for any $S \in \mathcal{S}^d$, by $\|S\|_2^2 = \sum_{i,j \in \mathbb{N}_d} S_{ij}^2$. Secondly, the closed convex sets are respectively given by $\mathcal{Q}_1 = \{u = (u_\tau : \tau \in \mathcal{T}) \in [0,1]^T\}$ and $\mathcal{Q}_2 = \{M \in \mathcal{S}_+^d : \mathbf{Tr}(M) \leq \sqrt{T/\gamma}\}$. Then, define the proxy-function $d_2(M) = \|M\|_2$. Consequently, the proxy-function $d_2(\cdot)$ is strongly convex on $\mathcal{Q}_2$ with convexity parameter $\sigma_2 = 2$. Finally, for any $\tau = (i, j, k) \in \mathcal{T}$, let

$X_\tau = x_{jk}x_{jk}^\top - x_{ij}x_{ij}^\top$. In addition, we replace the variable $x$ by $M$ and $\widehat{\phi}(u) = -\sum_{\tau \in \mathcal{T}} u_\tau$ in (12), $\hat{f}(M) = \gamma(\mathbf{Tr}(M))^2$. Finally, define the linear operator $A : \mathbb{R}^T \to (\mathcal{S}^d)^*$, for any $u \in \mathbb{R}^T$, by

$$Au = \sum_{\tau \in \mathcal{T}} u_\tau X_\tau. \tag{14}$$

With the above preparations, the saddle representation (7) exactly matches the special structure (11) which can be approximated by problem (12) with $\mu$ sufficiently small. The norm of the linear operator $A$ can be estimated as follows.

**Lemma 2.** *Let the linear operator $A$ be defined as above, then $\|A\|_{1,2} \leq \left(\sum_{\tau \in \mathcal{T}} \|X_\tau\|_2^2\right)^{\frac{1}{2}}$ where, for any $M \in \mathcal{S}^d$, $\|M\|_2$ denotes the Frobenius norm of $M$.*

*Proof.* For any $u \in \mathcal{Q}_1$ and $M \in \mathcal{S}^d$, we have that

$$\begin{aligned}
\mathbf{Tr}\left(\left(\sum_{\tau \in \mathcal{T}} u_\tau X_\tau\right)M\right) &\leq \left(\sum_{\tau \in \mathcal{T}} u_\tau \|X_\tau\|_2\right)\|M\|_2 \\
&\leq \|M\|_2\left(\sum_{\tau \in \mathcal{T}} u_\tau^2\right)^{\frac{1}{2}}\left(\sum_{\tau \in \mathcal{T}} \|X_\tau\|_2^2\right)^{\frac{1}{2}} = \|M\|_2\|u\|_1\left(\sum_{\tau \in \mathcal{T}} \|X_\tau\|_2^2\right)^{\frac{1}{2}}.
\end{aligned}$$

Combining the above inequality with the definition that $\|A\|_{1,2} = \max\{\mathbf{Tr}\left(\left(\sum_{\tau \in \mathcal{T}} u_\tau X_\tau\right)M\right) : \|u\|_1 = 1, \|M\|_2 = 1\}$ yields the desired result. $\qquad\square$

We now can adapt the smooth optimization [17, Section 3 and Theorem 3] to solve the smooth approximation formulation (12) for metric learning. To this end, let the proxy-function $d$ in $\mathcal{Q}_1$ be the standard Euclidean norm i.e. for some $u^{(0)} \in \mathcal{Q}_1 \subseteq \mathbb{R}^T$, $d(u) = \|u - u^{(0)}\|^2$. The smooth optimization pseudo-code for problem (7) (equivalently problem (4)) is outlined in Table 1. One can stop the algorithm by monitoring the relative change of the objective function or change in the dual gap.

The efficiency of Nesterov's smooth optimization largely depends on Steps 2, 3, and 4 in Table 1. Steps 3 and 4 can be solved straightforward where $z^{(t)} = \min(\max(0, u^{(t)} - \nabla\phi_\mu(u^{(t)})/L), 1)$ and $v^{(t)} = \min(\max(0, u^{(0)} - \sum_{i=0}^t (i+1)\nabla\phi_\mu(u^{(i)})/2L), 1)$. The solution $M_\gamma(u)$ in Step 2 involves the following problem

$$M_\mu(u) = \arg\max\{\langle \sum_{\tau \in \mathcal{T}} u_\tau X_\tau, M \rangle - \gamma(\mathbf{Tr}(M))^2 - \mu\|M\|_2^2 : M \in \mathcal{Q}_2\}. \tag{15}$$

The next lemma shows it can be efficiently solved by quadratic programming (QP).

**Lemma 3.** *Problem (15) is equivalent to the following*

$$s^* = \arg\max\left\{\sum_{i \in \mathbb{N}_d} \lambda_i s_i - \gamma(\sum_{i \in \mathbb{N}_d} s_i)^2 - \mu\sum_{i \in \mathbb{N}_d} s_i^2 : \sum_{i \in \mathbb{N}_d} s_i \leq \sqrt{T/\gamma}, \text{ and } s_i \geq 0 \ \forall i \in \mathbb{N}_d\right\} \tag{16}$$

*where $\lambda = (\lambda_1, \ldots, \lambda_d)$ are the eigenvalues of $\sum_{t \in \mathcal{T}} u_t X_t$. Moreover, if we denotes the eigen-decomposition $\sum_{t \in \mathcal{T}} u_t X_t$ by $\sum_{t \in \mathcal{T}} u_t X_t = U diag(\lambda) U^\top$ with some $U \in \mathcal{O}^d$ then the optimal solution of problem (15) is given by $M_\mu(u) = U diag(s^*) U^\top$.*

*Proof.* We know from Von Neumann's inequality (see [14] or [4, Page 10]), for all $X, Y \in \mathcal{S}^d$, that $\mathbf{Tr}(XY) \leq \sum_{i \in \mathbb{N}_d} \lambda_i(X)\lambda_i(Y)$ where $\lambda_i(X)$ and $\lambda_i(Y)$ are the eigenvalues of $X$ and $Y$ in non-decreasing order, respectively. The equality is attained whenever $X = U diag(\lambda(X))U^\top$, $Y = U diag(\lambda(Y))U^\top$ for some $U \in \mathcal{O}^d$. The desired result follows by applying the above inequality with $X = \sum_{\tau \in \mathcal{T}} u_\tau X_\tau$ and $Y = M$. $\qquad\square$

It was shown in [17, Theorem 3] that the iteration complexity is of $\mathcal{O}(1/\varepsilon)$ for finding a $\varepsilon$-optimal solution if we choose $\mu = \mathcal{O}(\varepsilon)$. This is usually much better than the standard sub-gradient descent method with iteration complexity typically $\mathcal{O}(1/\varepsilon^2)$. As listed in Table 1, the complexity for each iteration mainly depends on the eigen-decomposition on $\sum_{t \in \mathbb{N}_t} u_t X_t$ and the quadratic programming to solve problem (15) which has complexity $\mathcal{O}(d^3)$. Hence, the overall iteration complexity of the smooth optimization approach for sparse metric learning is of the order $\mathcal{O}(d^3/\varepsilon)$ for finding an $\varepsilon$-optimal solution. As a final remark, the Lipschitz given by the $L = \frac{1}{2\mu}\sum_\tau \|X_\tau\|^2$ could be too loose in reality. One can use the line search scheme [15] to further accelerate the algorithm.

# 4  Experiments

In this section we compared our proposed method with four other methods including (1) the **LMNN** method [23], (2) the Sparse Metric Learning via Linear Programming (**SMLlp**) [20], (3) the information-theoretic approach for metric learning (**ITML**) [9], and (4) the Euclidean distance based k-Nearest Neighbor (KNN) method (called **Euc** for brevity). We also implemented the iterative sub-gradient descent algorithm [24] to solve the proposed framework (4) (called **SMLgd**) in order to evaluate the efficiency of the proposed smooth optimization algorithm **SMLsm**. We try to exploit all these methods to learn a good distance metric and a KNN classifier is used to examine the performance of these different learned metrics.

The comparison is done on four benchmark data sets: Wine, Iris, Balance Scale, and Ionosphere, which were obtained from the UCI machine learning repository. We randomly partitioned the data sets into a training and test sets by using a ratio $0.85$. We then trained each approach on the training set, and performed evaluation on the test sets. We repeat the above process 10 times and then report the averaged result as the final performance. All the approaches except the Euclidean distance need to define a triplet set $\mathcal{T}$ before training. Following [20], we randomly generated 1500 triplets for SMLsm, SMLgd, SMLlp, and LMNN. The number of nearest neighbors was adapted via cross validation for all the methods in the range of $\{1, 3, 5, 7\}$. The trade-off parameter for SMLsm, SMLgd, SMLlp, and LMNN was also tuned via cross validation from $\{10^{-5}, 10^{-4}, 10^{-3}, 10^{-2}, 10^{-1}, 10^0, 10^1, 10^2\}$.

The first part of our evaluations focuses on testing the learning accuracy. The result can be seen in Figure 1 (a)-(d) respectively for the four data sets. Clearly, the proposed SMLsm demonstrates best performance. Specifically, SMLsm outperforms the other four methods in Wine and Iris, while it ranks the second in Balance Scale and Ionosphere with slightly lower accuracy than the best method. SMLgd showed different results with SMLsm due to the different optimization methods, which we will discuss shortly in Figure 1 (i)-(l). We also report the dimension reduction Figure 1(e)-(h). It is observed that our model outputs the most sparse metric. This validates the advantages of our approach. That is, our method directly learns both an accurate and sparse distance metric simultaneously. In contrast, other methods only touch this topic marginally: SMLlp is not optimal, as they exploited the one-norm regularization term and also relaxed the learning problem; LMNN aims to learn a metric with a large-margin regularization term, which is not directly related to sparsity of the distance matrix. ITML and Euc do not generate a sparse metric at all. Finally, in order to examine the efficiency of the proposed smooth optimization algorithm, we plot the convergence graphs of SMLsm versus those of SMLgd in Figure 1(i)-(l). As observed, SMLsm converged much faster than SMLgd in all the data sets. SMLgd sometimes oscillated and may incur a long tail due to the non-smooth nature of the hinge loss. For some data sets, it converged especially slow, which can be observed in Figure (k) and (l).

# 5  Conclusion

In this paper we proposed a novel regularization framework for learning a sparse (low-rank) distance matrix. This model was realized by a mixed-norm regularization term over a distance matrix which is non-convex. Using its special structure, it was shown to be equivalent to a convex min-max (saddle) representation involving a trace norm regularization. Depart from the saddle representation, we successfully developed an efficient Nesterov's first-order optimization approach [16, 17] for our metric learning model. Experimental results on various datasets show that our sparse metric learning framework outperforms other state-of-the-art methods with higher accuracy and significantly smaller dimensionality. In future, we are planning to apply our model to large-scale datasets with higher dimensional features and use the line search scheme [15] to further accelerate the algorithm.

## Acknowledgements

The second author is partially supported by the Excellent SKL Project of NSFC (No.60723005), China. The first and third author is supported by EPSRC grant EP/E027296/1.

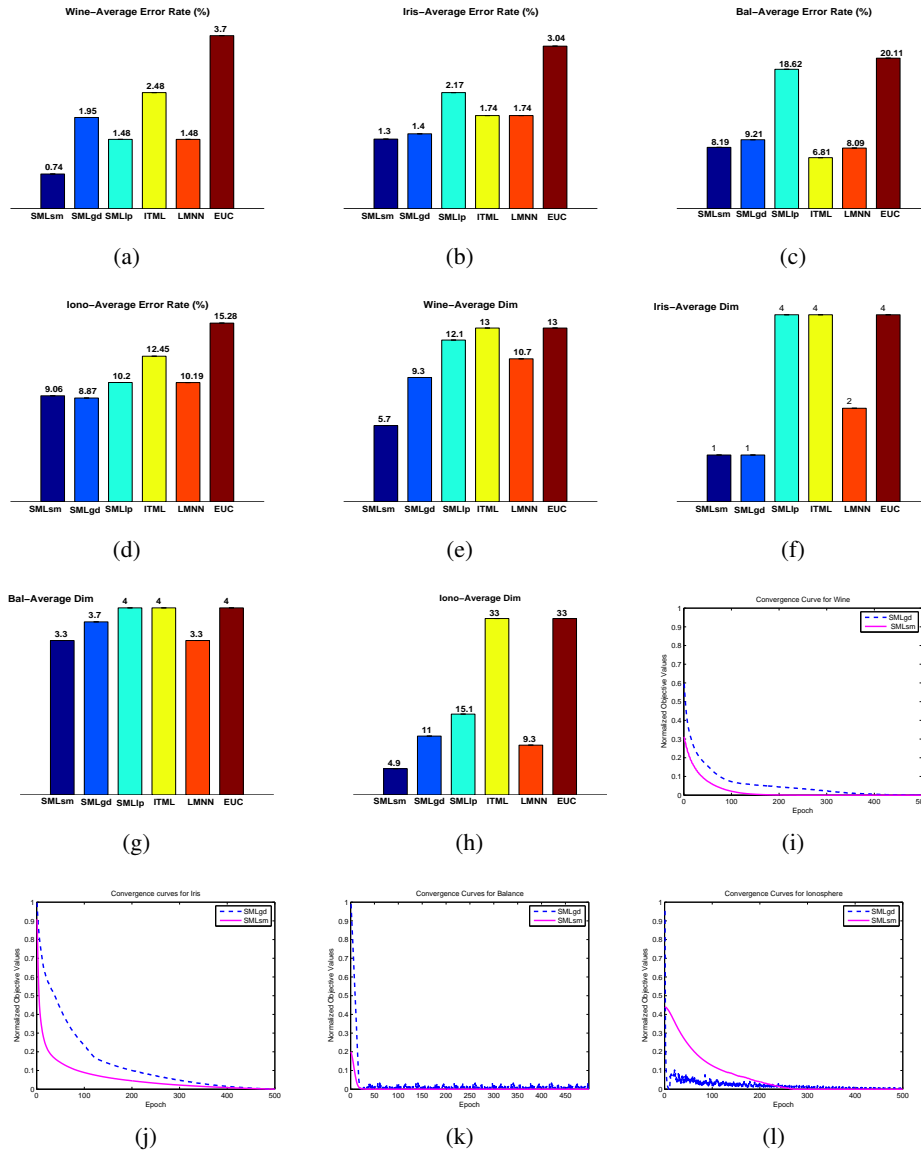

Figure 1: Performance comparison among different methods. Subfigures (a)-(d) present the average error rates; (e)-(h) plots the average dimensionality used in different methods; (i)-(l) give the convergence graph for the sub-gradient algorithm and the proposed smooth optimization algorithm.

# References

[1] A. Argyriou, T. Evgeniou, and M. Pontil. Multi-task feature learning. *NIPS*, 2007.

[2] A. Argyriou, C. A. Micchelli, M. Pontil, and Y. Ying. A spectral regularization framework for multi-task structure learning. *NIPS*, 2008.

[3] F. R. Bach. Consistency of trace norm minimization. *J. of Machine Learning Research*, **9**: 1019–1048, 2008.

[4] J. M. Borwein and A. S. Lewis. *Convex Analysis and Nonlinear Optimization: Theory and Examples*. CMS Books in Mathematics. Springer, 2005.

[5] S. Boyd and L . Vandenberghe. *Convex optimization*. Cambridge University Press, 2004.

[6] J. F. Bonnans and A. Shapiro. Optimization problems with perturbation: A guided tour. *SIAM Review*, **40**:202–227 ,1998.

[7] A. Bar-Hillel, T. Hertz, N. Shental, and D. Weinshall. Learning a mahalanobis metric from equivalence constraints. *J. of Machine Learning Research*, **6**: 937-965, 2005.

[8] S. Chopra, R. Hadsell, and Y. LeCun. Learning a similarity metric discriminatively with application to face verification. *CVPR*, 2005.

[9] J. Davis, B. Kulis, P. Jain, S. Sra, and I. Dhillon. Information-theoretic metric learning. *ICML*, 2007.

[10] G. M. Fung, O. L. Mangasarian, and A. J. Smola. Minimal kernel classifiers. *J. of Machine Learning Research*, **3**: 303–321, 2002.

[11] A. Globerson, S. Roweis, Metric learning by collapsing classes. *NIPS*, 2005.

[12] J. Goldberger, S. Roweis, G. Hinton, and R. Salakhutdinov. Neighbourhood component analysis. *NIPS*, 2004.

[13] T. Hastie, R.Tibshirani, and Robert Friedman. *The Elements of Statistical Learning*. Springer-Verlag New York, LLC, 2003.

[14] R.A. Horn and C.R. Johhnson. *Topics in Matrix Analysis*. Cambridge University Press, 1991.

[15] A. Nemirovski. *Efficient methods in convex programming*. Lecture Notes, 1994.

[16] Y. Nesterov. *Introductory Lectures on Convex Optimization: A Basic Course*. Springer, 2003.

[17] Y. Nesterov. Smooth minimization of non-smooth functions. *Mathematical Programming*, 103:127-152, 2005.

[18] Obozinski, B. Taskar, and M. I. Jordan. Joint covariate selection and joint subspace selection for multiple classification problems. *Statistics and Computing*. In press, 2009.

[19] J. D. M. Rennie, and N. Srebro. Fast maximum margin matrix factorization for collaborative prediction. *ICML*, 2005.

[20] R. Rosales and G. Fung. Learning sparse metrics via linear programming. *KDD*, 2006.

[21] N. Srebro, J.D. M. Rennie, and T. S. Jaakkola. Maximum-margin matrix factorization. *NIPS*, 2005.

[22] L. Torresani and K. Lee. Large margin component analysis. *NIPS*, 2007.

[23] K. Q. Weinberger, J. Blitzer, and L. Saul. Distance metric learning for large margin nearest neighbour classification. *NIPS*, 2006.

[24] K. Q. Weinberger and L. K. Saul. Fast solvers and efficient implementations for distance metric learning. *ICML*, 2008.

[25] E. Xing, A. Ng, M. Jordan, and S. Russell. Distance metric learning with application to clustering with side information. *NIPS*, 2002.

[26] L. Yang and R. Jin. Distance metric learning: A comprehensive survey. In *Technical report, Department of Computer Science and Engineering, Michigan State University*, 2007.

